# Spiking and saturating dendrites differentially expand single neuron computation capacity.

**Romain Cazé**
INSERM U960, Paris Diderot, Paris 7, ENS
29 rue d'Ulm, 75005 Paris
romain.caze@ens.fr

**Mark Humphries**
INSERM U960; University of Manchester
29 rue d'Ulm, 75005 Paris; UK
mark.humphries@manchester.ac.uk

**Boris Gutkin**
INSERM U960, CNRS, ENS
29 rue d'Ulm, 75005 Paris
boris.gutkin@ens.fr

## Abstract

The integration of excitatory inputs in dendrites is non-linear: multiple excitatory inputs can produce a local depolarization departing from the arithmetic sum of each input's response taken separately. If this depolarization is bigger than the arithmetic sum, the dendrite is spiking; if the depolarization is smaller, the dendrite is saturating. Decomposing a dendritic tree into independent dendritic spiking units greatly extends its computational capacity, as the neuron then maps onto a two layer neural network, enabling it to compute linearly non-separable Boolean functions (lnBFs). How can these lnBFs be implemented by dendritic architectures in practise? And can saturating dendrites equally expand computational capacity? To address these questions we use a binary neuron model and Boolean algebra. First, we confirm that spiking dendrites enable a neuron to compute lnBFs using an architecture based on the disjunctive normal form (DNF). Second, we prove that saturating dendrites as well as spiking dendrites enable a neuron to compute lnBFs using an architecture based on the conjunctive normal form (CNF). Contrary to a DNF-based architecture, in a CNF-based architecture, dendritic unit tunings do not imply the neuron tuning, as has been observed experimentally. Third, we show that one cannot use a DNF-based architecture with saturating dendrites. Consequently, we show that an important family of lnBFs implemented with a CNF-architecture can require an exponential number of saturating dendritic units, whereas the same family implemented with either a DNF-architecture or a CNF-architecture always require a linear number of spiking dendritic units. This minimization could explain why a neuron spends energetic resources to make its dendrites spike.

## 1   Introduction

Recent progress in voltage clamp techniques has enabled the recording of local membrane voltage in dendritic branches, and this greatly changed our view of the potential for single neuron computation. Experiments have shown that when the local dendritic membrane potential reaches a given threshold a dendritic spike can be elicited [4, 13]. Based on this type of local dendritic non-linearity, it has been suggested that a CA1 hippocampal pyramidal neuron comprises multiple independent non-linear spiking units, summating at the soma, and is thus equivalent to a two layer artificial neural network [12]. This idea is attractive, because this type of feed-forward network can implement any Boolean function, in particular linearly non-separable Boolean functions (lnBFs), and thus radically

extends the computational power of a single neuron. By contrast, a seminal neuron model, the McCulloch & Pitts unit [10], is restricted to linearly separable Boolean functions.

However attractive this idea, it requires additional investigation. Indeed, spiking dendritic unit may enable the computation of lnBFs using an architecture, suggested in [9], where the dendritic tuning implies the neuron tuning (see also Proposition 1). This relation between dendritic and neuron tuning has not been confirmed experimentally; on the contrary it has been shown in vivo that dendritic tuning does not imply the neuron tuning [6]: calcium imaging in vivo has shown that the local calcium signal in dendrites can maximally increase for visual inputs whereas that do not trigger somatic spiking. We resolve this first issue here by showing how one can implement lnBFs with spiking dendritic units, whose tunings do not formally imply the somatic tuning.

Moreover, the idea of a neuron implementing a two-layer network is based on spiking dendrites. Dendritic non-linearities have a variety of shapes, and many neuron types may not have the capacity to generate dendritic spikes. By contrast, all dendrites can saturate [1, 16, 2]. For instance, gluta-mate uncaging on cerebellar stellate cell dendrites and simultaneous somatic voltage recording of these interneurons shows that multiple excitatory inputs on the same dendrite result in a somatic depolarization smaller than the arithmetic sum of the quantal depolarizations [1]. This type of non-linearity has been predicted from Rall's work [7], a model which explains saturation by an increase in membrane conductance and a decrease in driving force. It is unknown whether local dendritic saturation can also enhance the general computational capacity of a single neuron in the same way as local dendritic spiking – but, if so, this would make plausible the implementation of lnBFs in potentially any type of neuron. In the present study we show that saturating dendritic units do also enable the computation of lnBFs (see Proposition 2).

One can wonder why some dendrites support metabolically-expensive spiking if dendritic saturation is sufficient to compute all Boolean functions. We tackle this issue in the second part of our study. We show that a family of positive lnBFs may require an exponentially growing number of saturat-ing dendritic units when the number of input variables grow linearly, whereas the same family of Boolean functions requires a linearly growing number of spiking dendritic units. Consequently den-dritic spikes may minimize the number of units necessary to implement *all* Boolean functions. Thus, as the number of independent units – spiking or saturating – in a dendrite remains an open question [5], but potentially small [14], it may turn out that certain Boolean functions are only implementable using spiking dendrites.

## 2 Definitions

### 2.1 The binary two stage neuron

We introduce here a neuron model analogous to [12]. Our model is a binary two stage neuron, where $X$ is a binary input vector of length $n$ and $y$ is a binary variable modelling the neuron output. First, inputs sum locally within each dendritic unit $j$ given a local weight vector $W_j$; then they pass though a local transfer function $F_j$ accounting for the dendritic non-linear behavior. Second, outputs of the $d$ dendritic subunits sum at the soma and passes though the somatic transfer function $F_0$. $F_0$ is a spiking transfer function whereas $F_j$ are either spiking or saturating transfer functions, these functions are described in the next section and are displayed on Figure 1A. Formally, the output $y$ is computed with the following equation:

$$y = F_0\Big( \sum_{j=1}^{d} F_j(W_j.X) \Big)$$

### 2.2 Sub-linear and supra-linear transfer functions

A transfer function $F$ takes as input a local weighted linear sum $x$ and outputs $F(x)$; this output depends on the type of transfer function: spiking or saturating, and on a single positive parameter $\Theta$ the threshold of the transfer function. The two types of transfer functions are defined as follows:

**Definition 1. Spiking transfer function**

$$F_{spk}(x) = \begin{cases} 1 & if\ x \geq \Theta \\ 0 & otherwise \end{cases}$$

Table 1: Two examples of positive Boolean functions of 4 variables

| $x_1$ | 0 | 1 | 0 | 1 | 0 | 1 | 0 | 1 | 0 | 1 | 0 | 1 | 0 | 1 | 0 | 1 |
|---|---|---|---|---|---|---|---|---|---|---|---|---|---|---|---|---|
| $x_2$ | 0 | 0 | 1 | 1 | 0 | 0 | 1 | 1 | 0 | 0 | 1 | 1 | 0 | 0 | 1 | 1 |
| $x_3$ | 0 | 0 | 0 | 0 | 1 | 1 | 1 | 1 | 0 | 0 | 0 | 0 | 1 | 1 | 1 | 1 |
| $x_4$ | 0 | 0 | 0 | 0 | 0 | 0 | 0 | 0 | 1 | 1 | 1 | 1 | 1 | 1 | 1 | 1 |
| $g(x_1, x_2, x_3, x_4)$ | 0 | 0 | 0 | 1 | 0 | 0 | 0 | 1 | 0 | 0 | 0 | 1 | 1 | 1 | 1 | 1 |
| $h(x_1, x_2, x_3, x_4)$ | 0 | 0 | 0 | 0 | 0 | 1 | 1 | 1 | 0 | 1 | 1 | 1 | 0 | 1 | 1 | 1 |

**Definition 2.  Saturating transfer function**

$$F_{sat}(x) = \begin{cases} 1 & \text{if } x \geq \Theta \\ x/\Theta & \text{otherwise} \end{cases}$$

The difference between a spiking and a saturating transfer function is that $F_{spk}(x) = 0$ whereas $F_{sat}(x) = x/\Theta$ if $x$ is below $\Theta$. To formally characterize this difference we define here sub-linearity and supra-linearity of a transfer function $F$ on a given interval $I$. These definitions are similar to the well-known notions of concavity and convexity:

**Definition 3.** *F is supra-linear on I if and only if $F(x_1 + x_2) > F(x_1) + F(x_2)$ for at least one $(x_1, x_2) \in I^2$*

*F is sub-linear on I if and only if $F(x_1 + x_2) < F(x_1) + F(x_2)$ for at least one $(x_1, x_2) \in I^2$*

*F is strictly sub-linear (resp. supra-linear) on I if it is sub-linear (resp. supra-linear) but not supra-linear (resp. sub-linear) on I.*

*Note that these definitions also work when using $n$-tuples instead of couples on the interval (useful in Lemma 3).*

Note that whenever $\Theta > 0$, $F_{spk}$ is both supra and sub-linear on $I = [0, +\infty[$ whereas $F_{sat}$ is strictly sub-linear on the same interval. $F_{sat}$ is not supra-linear on $I$ because $F_{sat}(x_1 + x_2) \leq F_{sat}(x_1) + F_{sat}(x_2)$ for all $(x_1, x_2) \in I^2$, by definition of $F_{sat}$. Moreover, $F_{sat}$ is sub-linear on $I$ because $F_{sat}(a + b) = 1$ and $F_{sat}(a) + F_{sat}(b) = 2$ for at least one $(a, b) \in I^2$ such that $a \geq \Theta$ and $b \geq \Theta$. All in all, $F_{sat}$ is strictly sub-linear on $I$.

Similarly to $F_{sat}$, $F_{spk}$ is sub-linear on $I$ because $F_{spk}(a + b) = 1$ and $F_{spk}(a) + F_{spk}(b) = 2$ for at least one $(a, b) \in I^2$ such that $a \geq \Theta$ and $b \geq \Theta$. Moreover, $F_{spk}$ is supra-linear because $F_{spk}(c + d) = 1$ and $F_{spk}(c) + F_{spk}(d) = 0$ for at least one $(c, d)$ such that $c < \Theta$ and $d < \Theta$ but $c + d \geq \Theta$. All in all, $F_{spk}$ is both sub-linear and supra-linear.

## 2.3   Boolean Algebra

In order to study the range of possible input-output mappings implementable by a two stage neurons we use Boolean functions, which can efficiently and formally describe all binary input-output mappings. Let us recall the definition of this extensively studied mathematical object [3, 17]:

**Definition 4.** *A Boolean function of $n$ variables is a function on $\{0, 1\}^n$ into $\{0, 1\}$, where $n$ is a positive integer.*

In Table.1 the truth table for two Boolean functions $g$ and $h$ is presented. These Boolean functions are fully and uniquely defined by their truth table. Both $g$ and $h$ are positive lnBFs (see chapter 9 of [3] for an extensive study of linear separability); because of its importance we recall the definition of positive Boolean functions:

**Definition 5.** *Let $f$ be a Boolean function on $\{0, 1\}^n$. $f$ is positive if and only if $f(X) \geq f(Z)$ $\forall (X, Z) \in \{0, 1\}^n$ such that $X \geq Z$ (meaning that $\forall i : x_i \geq z_i$)*

We also recall the notion of implication as it is important to observe that a dendritic input-output function (or *tuning*) may or not imply the neuron's input-output function:

**Definition 6.** *Let $f$ and $g$ be two Boolean functions.*

$$f \text{ implies } g \iff f(X) = 1 \implies g(X) = 1 \; \forall X \in \{0,1\}^n$$

As will become clear, we can treat each dendritic unit as computing its own Boolean function on its inputs: for a unit's output to imply the whole neuron's output then means that if a unit outputs a 1, then the neuron outputs a 1.

In order to describe positive Boolean functions, it is useful to decompose them into positive *terms* and positive *clauses*:

**Definition 7.** *Let $X^{(j)}$ be a tuple of $k < n$ positive integers referencing the different variables present in a term or a clause.*

*A positive term $j$ is a conjunction of variables written as $T_j(X) = \bigwedge\limits_{i \in X^{(j)}} x_i$.*

*A positive clause $j$ is a disjunction of variables written as $C_j(X) = \bigvee\limits_{i \in X^{(j)}} x_i$.*

*A term or (resp. clause) is prime if it is not implied by (resp. does not imply) any other term (resp. clause) in a disjunction (resp. conjunction) of multiple terms (resp. clauses).*

These terms and clauses can then define the Disjunctive or Conjunctive Normal Form (DNF or CNF) expression of a Boolean function $f$, particularly:

**Definition 8.** **A complete positive DNF** *is a disjunction of prime positive terms $T$:*

$$DNF(f) := \bigvee_{T_j \in T} \left( \bigwedge_{i \in X^{(j)}} x_i \right)$$

**Definition 9.** **A complete positive CNF** *is a conjunction of prime positive clauses $C$:*

$$CNF(f) := \bigwedge_{C_j \in C} \left( \bigvee_{i \in X^{(j)}} x_i \right)$$

It has been shown that all positive Boolean functions can be expressed as a positive complete DNF ([3] Theorem 1.24); similarly all positive Boolean functions can be expressed as a positive complete CNF. These complete positive DNF or CNF are the shortest possible DNF or CNF descriptions of positive Boolean functions. To clarify all these definitions let us introduce a series of examples build around $g$ and $h$.

**Example 1.** *Let us take $X^{(1)} = (1,2)$ and $X^{(2)} = (3,4)$. These tuples define two positive terms $T_1(X) = x_1 \wedge x_2$ where $T_1(X) = 1$ only when $x_1 = 1$ and $x_2 = 1$ and $T_1(X) = 0$ otherwise; similarly $T_2(X) = x_3 \wedge x_4$ where $T_2(X) = 1$ only when $x_3 = 1$ and $x_4 = 1$. These tuples can also define two positive clauses $C_1(X) = x_1 \vee x_2$ where $C_1(X) = 1$ as soon as $x_1 = 1$ or $x_2 = 1$, and similarly $C_2(X) = x_3 \vee x_4$ where $C_2(X) = 1$ as soon as $x_3 = 1$ or $x_4 = 1$. In the disjunction of terms $T_1 \vee T_2$ the terms are prime because $T_1(X) = 1$ is not implied by $T_2(X) = 1$ for all $X$ (and vice-versa). Similarly in the conjunction of clauses $C_1 \wedge C_2$ the clauses are prime because $C_1(X) = 1$ does not imply that $C_2(X) = 1$ for all $X$ (and vice-versa). $T_1 \vee T_2$ is the complete positive DNF expression of $g$; alternatively $C_1 \wedge C_2$ is the complete positive CNF expression of $h$. The truth tables of $g$ and $h$ are displayed in Table 1*

## 3   Results

We first prove here that a two stage neuron with a sufficient number of only spiking or only saturating dendritic units can implement all positive Boolean functions, particularly lnBFs like $g$ and $h$, whereas a classic McCulloch & Pitts unit is restricted to linearly separable Boolean functions. Moreover, we present two construction architectures for building a two stage neuron implementing a positive Boolean function based on its complete DNF or CNF expression. Finally we show that the DNF-based architecture is only possible with spiking dendritic units and not with saturating dendritic units.

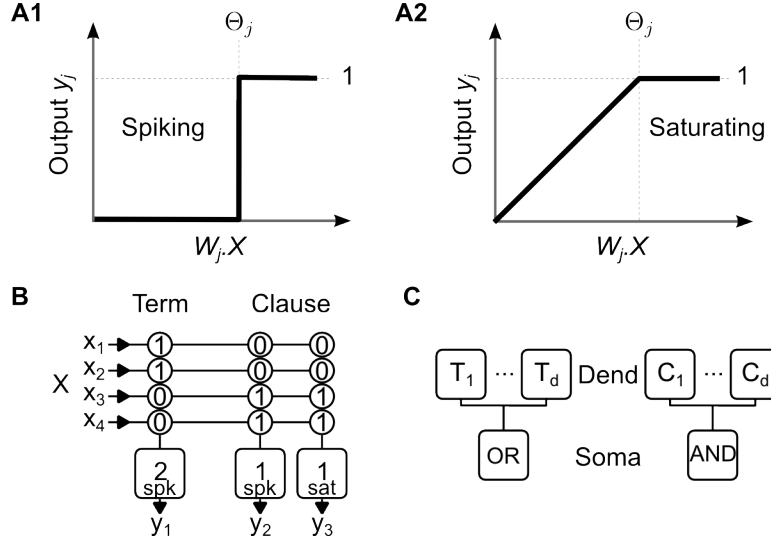

Figure 1: **Modeling dendritic spikes, dendritic saturations, and their impact on computation capacity** (A) Two types of transfer functions for a unit $j$ with a normalized height to 1 and a variable threshold $\Theta_j$. The input is the local weighted sum $W_j.X$ and the output is $y_j$ (A1) A spiking transfer function models somatic spikes and dendritic spikes (A2) A saturating transfer function models dendritic saturations (B) From left to right, a unit implementing the term $T(X) = x_1 \lor x_2$, and two units implementing the clause $C(X) = x_3 \lor x_4$, in circles are synaptic weights and in squares are threshold and the type of transfer function (spk:spiking, sat:saturating) (C) Two architectures to implement all positive Boolean functions in a two stage neuron, the $d$ dendritic units correspond to all terms of a DNF (left) or to all the clauses of a CNF (right), the somatic unit respectively implements an AND or an OR logic operation

## 3.1 Computation of positive Boolean functions using non-linear dendritic units

**Lemma 1.** *A two stage neuron with non-negative synaptic weights and increasing transfer functions necessarily implements positive Boolean functions*

*Proof.* Let $f$ be the Boolean function representing the input-output mapping of a two stage neuron, and two binary vectors $X$ and $Z$ such that $X \geq Z$. We have $\forall j \in \{1, 2, \ldots, d\}$ non-negative local weights $w_{i,j} \geq 0$, thus for a given dendritic unit $j$ we have:

$$w_{i,j}x_i \geq w_{i,j}z_i.$$

We can sum inequalities for all $i$, and $F_j$ are increasing transfer functions thus:

$$F_j(W_j.X) \geq F_j(W_j.Z).$$

We can sum the $d$ inequalities corresponding to every dendritic unit, and $F_0$ is an increasing transfer function thus:

$$f(X) \geq f(Z).$$

$\square$

**Lemma 2.** *A term (resp. a clause) can be implemented by a unit with a supra-linear (resp. sub-linear) transfer function*

*Proof.* We need to provide the parameter sets of a transfer function implementing a term (resp. a clause) with the constraint that the transfer function is supra-linear (resp. sub-linear). Indeed, a supra-linear transfer function (like the spiking transfer function) with the parameter set $w_i = 1$ if $i \in X^{(j)}$ and $w_i = 0$ otherwise and $\Theta = \text{card}(X^{(j)})$ implements the term $T_j$. A sub-linear transfer function (like the saturating transfer function) with the parameter set $w_i = 1$ if $i \in X^{(j)}$ and $w_i = 0$ otherwise and $\Theta = 1$ implements the clause $C_j$. These implementation are illustrated by examples in Figure 1B $\square$

**Lemma 3.** *A term (resp. a clause) cannot be implemented by a unit with a strictly sub-linear (resp. supra-linear) transfer function*

*Proof.* We prove this lemma for a term, the proof is similar for a clause. Let $T_j$ be the term defined by $X^{(j)}$, with $\text{card}(X^{(j)}) \geq 2$. First, for all input vectors $X$ such that $x_i = 1$ with $i \in X^{(j)}$ and $x_{k \neq i} = 0$ then $T_j(X) = 0$ implying that $F(W.X) = F(w_i x_i) = 0$. One can sum all these elements to obtain the following equality $\sum_{i \in X^{(j)}} F(w_i x_i) = 0$. Second, for all input vectors $X$ such that $x_i = 1$ for all $i \in X^{(j)}$ then $T_j(X) = 1$ implying that $F\left( \sum_{i \in X^{(j)}} w_i x_i \right) = 1$. Putting the two pieces together we obtain:

$$F\left( \sum_{i \in X^{(j)}} w_i x_i \right) > \sum_{i \in X^{(j)}} F(w_i x_i)$$

This inequality shows that the tuple of points $(w_i x_i | i \in X^{(j)})$ defining a term must have $F$ supra-linear; therefore, by Definition 2, $F$ cannot be both strictly sub-linear and implement a term. □

Using these Lemmas we show the possible and impossible implementation architectures of positive Boolean functions in two-layer neuron models using either spiking or saturating dendritic units.

**Proposition 1.** *A two stage neuron with non-negative synaptic weights and a sufficient number of dendritic units with spiking transfer functions can implement only and all positive Boolean functions based on their positive complete DNF*

*Proof.* A two stage neuron can only compute positive Boolean functions (Lemma 1). All positive Boolean functions can be expressed as a positive complete DNF; because a spiking dendritic unit has a supra-linear transfer function it can implement all possible terms (Lemma 2). Therefore a two stage neuron model without inhibition can implement only and all positive Boolean functions with as many dendritic units as there are terms in the functions' positive complete DNF. This architecture is represented on Figure 1C (left). □

Informally, this simply means that a dendrite is a pattern detector: if a pattern is present in the input then the dendritic unit elicits a dendritic spike. This architecture has been repeatedly invoked by theoreticians [8] and experimentalists ([9] in supplementary material) to suggest that dendritic spikes increase a neuron's computational capacity. With this architecture, however, the dendritic transfer function, if it is viewed as a Boolean function, formally implies the neuron's input-output mapping. This has not been confirmed experimentally yet.

**Proposition 2.** *A two stage neuron with non-negative synaptic weights and a sufficient number of dendritic units with spiking or saturating transfer functions can implement only and all positive Boolean functions based on their positive complete CNF*

*Proof.* A two stage neuron can only compute positive Boolean functions (Lemma 1). All positive Boolean functions can be expressed as a positive complete CNF; because a spiking or a saturating dendritic unit has a sub-linear transfer function they both can implement all possible clauses (Lemma 2). Therefore a two stage neuron model without inhibition can implement only and all positive Boolean functions with as many dendritic units as there are clauses in the functions' positive complete CNF. This architecture is represented on Figure 1C (right). □

To our knowledge, this implementation architecture has not yet been proposed in the neuroscience literature. It shows that saturations can increase the computational power of a neuron as much as dendritic spikes. It also shows that another implementation architecture is possible using spiking dendritic units. Using this architecture, the dendritic units' transfer functions do not imply the somatic output. This independence of dendritic and somatic response to inputs has been observed in Layer 2/3 neurons [6].

**Proposition 3.** *A two stage neuron with non-negative synaptic weights and only dendritic units with saturating transfer functions cannot implement a positive Boolean function based on its complete DNF*

*Proof.* The transfer function of a saturating dendritic unit is strictly sub-linear, therefore this unit cannot implement a term (Lemma 3). □

This result suggests that spiking dendritic units are more flexible than saturating dendritic units; they allow the computation of Boolean functions through either DNF or CNF-based architectures (illustrated in Figure 2), whereas saturating units are restricted to CNF-based architectures.

### 3.2 Implementation of a family of positive lnBFs using either spiking or saturating dendrites

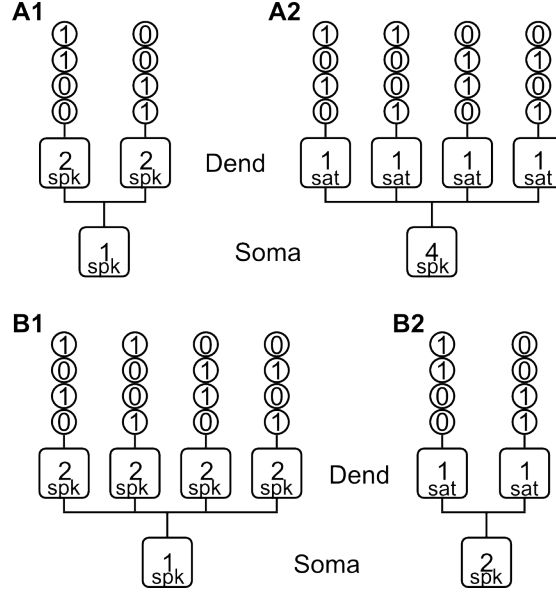

Figure 2: **Implementation of two linearly non-separable Boolean functions using CNF-based or DNF-based architectures**. Four parameter sets of two-stage neuron models: in circles are synaptic weights and in squares are threshold and the unit type (spk:spiking, sat:saturating). These parameter sets implement (A1/A2) $g$ or (B1/B2) $h$, two lnBFs depicted in Table 1 using: (A1/B1) a DNF-based architecture and spiking dendritic units only; (A2/B2) a CNF-based architecture and saturating dendritic units only.

The Boolean functions $g$ and $h$ form a family of Boolean functions we call feature binding problems in reference to [8]. In this section we show how this family can be implemented using either a DNF-based or CNF-based architecture. For some Boolean functions, the DNF and CNF grow at different rates as a function of the number of variables [3, 11]. This is the case when $g$ and $h$ are defined for $n$ input variables.

**Example 2.** *Let's define $g$ by the complete positive DNF expression $\phi$ :*

$$\phi(g(x_1, z_1, \ldots, x_n, z_n)) := x_1 z_1 \lor x_2 z_2 \lor \cdots \lor x_n z_n$$

*The same function $g$ has a unique complete positive CNF expression; let's call it $\psi$. The clauses of $\psi$ are exactly those elementary disjunctions of $n$ variables that involve one variable out of each of the pairs $\{x_1, z_1\}, \{x_2, z_2\}, \ldots, \{x_n, z_n\}$. Thus $\psi$ has $2^n$ clauses.*

**Example 3.** *Let's define $h$ by the complete positive CNF expression $\psi$:*

$$\psi(h(x_1, z_1, \ldots, x_n, z_n)) := (x_1 \lor z_1)(x_2 \lor z_2) \ldots (x_n \lor z_n)$$

*The same function $h$ has a unique complete positive DNF expression; let's call it $\phi$. The terms of $\phi$ are exactly those elementary conjunctions of $n$ variables that involve one variable out of each of the pairs $\{x_1, z_1\}, \{x_2, z_2\}, \ldots, \{x_n, z_n\}$. Thus $\phi$ has $2^n$ terms.*

Table 2 shows the number of necessary units for $g$ and $h$ depending on the chosen architecture. From Propositions 1 and 2, it is immediately clear that spiking dendritic units always give access to the

Table 2: Number of necessary units

| Boolean function | # of terms in DNF | # of clauses in CNF |
|---|---|---|
| $g$ | $n$ | $2^n$ |
| $h$ | $2^n$ | $n$ |

minimal possible two-stage neuron implementation. A neuron with spiking dendritic units can thus implement $g$ with $n$ units using DNF-based and $h$ with $n$ units using CNF-based architectures; but saturating units, restricted to CNF-based architectures, can only implement $h$ with $2^n$ units.

## 4 Discussion

The main result of our study is that dendritic saturations can play a computational role that is as important as dendritic spikes: saturating dendritic units enable a neuron to compute lnBFs (as shown in Proposition 2). The same Proposition shows that a neuron can compute lnBFs decomposed according to the CNF using spiking dendritic units; with this architecture, dendritic tuning does not imply the somatic tuning to inputs. Moreover, we demonstrated that an important family of lnBFs formed by $g$ and $h$ can be implemented in a two stage neuron using either spiking or saturating dendritic units. We also showed that lnBFs cannot be implemented in a two stage neuron using a DNF-based architecture with only dendritic saturating units (Proposition 3).

These results nicely separate the implications of saturating and spiking dendritic units in single neuron computation. On the one hand, spiking dendritic units are a more flexible basis for computation, as they can be employed in two different implementation architectures (Proposition 1 and 2) where dendritic tunings – the dendritic unit transfer functions – can imply or not the tuning of the whole neuron. The latter may explain why dendrites can have a tuning different from the whole neuron as has been observed in Layer 2/3 pyramidal cells of the visual cortex [6]. On the other hand, saturating dendritic units can enhance single neuron computation through implementing all positive Boolean functions (Proposition 3), while reducing the energetic costs associated with the active ion channels required for dendritic spikes [4, 13].

For an infinite number of dendritic units, saturating and spiking units lead to the same increase in computation capacity; for a finite number of dendritic units our results suggests that spiking dendritic units could have advantages over saturating dendritic units. In the second part of our study we showed that a family of lnBFs can be described by an expression containing an exponential or a linear number of elements. Namely, the lnBFs defined by $g$ or $h$ can be implemented with a linear number of spiking dendritic units whereas for $g$ a neuronal implementation using only saturations requires an exponential number of saturating dendritic units. Consequently, spiking dendritic units may allow the minimization of dendritic units necessary to implement this family of Boolean functions.

The Boolean functions $g$ and $h$ formalize feature binding problems [8] which are important and challenging computations (see [15] for review). Some single neuron solutions to feature binding problems have been proposed in [8], but restricted to DNF-based architectures; our results thus generalize and extend this study by proposing alternative CNF-based solutions. Moreover, we show that this alternative architecture enables the solution of an important family of binding problems with a linear number of spiking dendritic unit. Thus we have proposed more efficient solutions to a family of challenging computations.

Because of their elegance and simplicity stemming from Boolean algebra, we believe our results are applicable to more complex situations. They can be extended to continuous transfer functions, which are more biologically plausible; in this case the notion of sub-linearity and supra-linearity are replaced by concavity and convexity. Moreover, all the parameters used here for proofs and examples are integer-valued but the same proofs and examples are easily extendable to continuous steady-state rate models where parameters are real-valued. In conclusion, our results have a solid formal basis, moreover, they both explain recent experimental findings and suggest a new way to implement Boolean functions using saturating as well as spiking dendritic units.

# References

[1] T. Abrahamsson, L. Cathala, K. Matsui, R. Shigemoto, and D.A. DiGregorio. Thin Dendrites of Cerebellar Interneurons Confer Sublinear Synaptic Integration and a Gradient of Short-Term Plasticity. *Neuron*, 73(6):1159–1172, March 2012.

[2] S. Cash and R. Yuste. Linear summation of excitatory inputs by CA1 pyramidal neurons. *Neuron*, 22(2):383–394, February 1999.

[3] Y. Crama and P.L. Hammer. *Boolean Functions: Theory, Algorithms, and Applications (Encyclopedia of Mathematics and its Applications)*. Cambridge University Press, 2011.

[4] S. Gasparini, M. Migliore, and J.C. Magee. On the initiation and propagation of dendritic spikes in CA1 pyramidal neurons. *The Journal of Neuroscience*, 24(49):11046–11056, December 2004.

[5] M. Hausser and B.W. Mel. Dendrites: bug or feature? *Current Opinion in Neurobiology*, 13(3):372–383, June 2003.

[6] H. Jia, N.L. Rochefort, X. Chen, and A. Konnerth. Dendritic organization of sensory input to cortical neurons in vivo. *Nature*, 464(7293):1307–1312, 2010.

[7] C. Koch. *Biophysics of computation : information processing in single neurons*. Oxford University Press, New York, 1999.

[8] R. Legenstein and W. Maass. Branch-Specific Plasticity Enables Self-Organization of Nonlinear Computation in Single Neurons. *Journal of Neuroscience*, 31(30):10787–10802, July 2011.

[9] A. Losonczy, J.K. Makara, and J.C. Magee. Compartmentalized dendritic plasticity and input feature storage in neurons. *Nature*, 452(7186):436–441, March 2008.

[10] W.S. McCulloch and W. Pitts. A logical calculus of the ideas immanent in nervous activity. *Bulletin of mathematical biology*, 52(1-2):99–115; discussion 73–97, January 1943.

[11] P.B. Miltersen, J. Radhakrishnan, and I. Wegener. On converting CNF to DNF. *Theoretical computer science*, 347:325–335, November 2005.

[12] P. Poirazi, T. Brannon, and B.W. Mel. Pyramidal neuron as two-layer neural network. *Neuron*, 37(6):989–999, March 2003.

[13] A. Polsky, B.W. Mel, and J. Schiller. Computational subunits in thin dendrites of pyramidal cells. *Nature Neuroscience*, 7(6):621–627, June 2004.

[14] M.W.H. H Remme, M. Lengyel, and B.S. Gutkin. Democracy-independence trade-off in oscillating dendrites and its implications for grid cells. *Neuron*, 66(3):429–37, May 2010.

[15] A.L. Roskies. The Binding Problem. *Neuron*, 24:7–9, 1999.

[16] K. Vervaeke, A. Lorincz, Z. Nusser, and R.A. Silver. Gap Junctions Compensate for Sublinear Dendritic Integration in an Inhibitory Network. *Science*, 335(6076):1624–1628, March 2012.

[17] I. Wegener. *Complexity of Boolean Functions*. Wiley-Teubner, 1987.

